# Nonlinear directed acyclic structure learning with weakly additive noise models

**Robert E. Tillman**
Carnegie Mellon University
Pittsburgh, PA
rtillman@cmu.edu

**Arthur Gretton**
Carnegie Mellon University,
MPI for Biological Cybernetics
Pittsburgh, PA
arthur.gretton@gmail.com

**Peter Spirtes**
Carnegie Mellon University
Pittsburgh, PA
ps7z@andrew.cmu.edu

## Abstract

The recently proposed *additive noise model* has advantages over previous directed structure learning approaches since it (i) does not assume linearity or Gaussianity and (ii) can discover a unique DAG rather than its Markov equivalence class. However, for certain distributions, e.g. linear Gaussians, the additive noise model is invertible and thus not useful for structure learning, and it was originally proposed for the two variable case with a multivariate extension which requires enumerating all possible DAGs. We introduce *weakly additive noise models*, which extends this framework to cases where the additive noise model is invertible and when additive noise is not present. We then provide an algorithm that learns an equivalence class for such models from data, by combining a PC style search using recent advances in kernel measures of conditional dependence with local searches for additive noise models in substructures of the Markov equivalence class. This results in a more computationally efficient approach that is useful for arbitrary distributions even when additive noise models are invertible.

## 1 Introduction

Learning probabilistic graphical models from data serves two primary purposes: (i) finding compact representations of probability distributions to make inference efficient and (ii) modeling unknown data generating mechanisms and predicting causal relationships. Until recently, most constraint-based and score-based algorithms for learning directed graphical models from continuous data required assuming relationships between variables are linear with Gaussian noise. While this assumption may be appropriate in many contexts, there are well known contexts, such as fMRI images, where variables have nonlinear dependencies and data do not tend towards Gaussianity. A second major limitation of the traditional algorithms is they cannot identify a unique structure; they reduce the set of possible structures to an equivalence class which entail the same Markov properties. The recently proposed *additive noise model* [1] for structure learning addresses both limitations; by taking advantage of observed nonlinearity and non-Gaussianity, a unique directed acyclic structure can be identified in many contexts. However, it too suffers from limitations: (i) for certain distributions, e.g. linear Gaussians, the model is invertible and not useful for structure learning; (ii) it was originally proposed for two variables with a multivariate extension that requires enumerating all possible DAGs, which is super-exponential in the number of variables.

In this paper, we address the limitations of the additive noise model. We introduce *weakly additive noise models*, which have the advantages of additive noise models, but are still useful when the additive noise model is invertible and in most cases when additive noise is not present. Weakly additive noise models allow us to express greater uncertainty about the

data generating mechanism, but can still identify a unique structure or a smaller equivalence class in most cases. We also provide an algorithm for learning an equivalence class for such models from data that is more computationally efficient in the more than two variables case. Section 2 reviews the appropriate background; section 3 introduces weakly additive noise models; section 4 describes our learning algorithm; section 5 discusses some related research; section 6 presents some experimental results; finally, section 7 offers conclusions..

## 2  Background

Let $\mathcal{G} = \langle \mathcal{V}, \mathcal{E} \rangle$ be a directed acyclic graph (DAG), where $\mathcal{V}$ denotes the set of vertices and $E_{ij} \in \mathcal{E}$ denotes a directed edge $V_i \rightarrow V_j$. $V_i$ is a *parent* of $V_j$ and $V_j$ is a *child* of $V_i$. For $V_i \in \mathcal{V}$, $\mathbf{Pa}_{\mathcal{G}}^{V_i}$ denotes the parents of $V_i$ and $\mathbf{Ch}_{\mathcal{G}}^{V_i}$ denotes the children of $V_i$. The *degree* of $V_i$ is the number of edges with an endpoint at $V_i$. A *v-structure* is a triple $\langle V_i, V_j, V_k \rangle \subseteq \mathcal{V}$ such that $\{V_i, V_k\} \subseteq \mathbf{Pa}_{\mathcal{G}}^{V_j}$. A v-structure is *immoral*, or an *immorality*, if $E_{ik} \notin \mathcal{E}$ and $E_{ki} \notin \mathcal{E}$. A joint distribution $\mathcal{P}$ over variables corresponding to nodes in $\mathcal{V}$ is *Markov* with respect to $\mathcal{G}$ if $\mathbb{P}_{\mathcal{P}}(\mathcal{V}) = \prod_{V_i \in \mathcal{V}} \mathbb{P}_{\mathcal{P}} \left( V_i \mid \mathbf{Pa}_{\mathcal{G}}^{V_i} \right)$. $\mathcal{P}$ is *faithful* to $\mathcal{G}$ if every conditional independence true in $\mathcal{P}$ is entailed by the above factorization. A *partially directed acyclic graph* (PDAG) $\mathcal{H}$ for $\mathcal{G}$ is a *mixed graph*, i.e. consisting of directed and undirected edges, representing all DAGs *Markov equivalent* to $\mathcal{G}$, i.e. DAGs entailing exactly the same conditional independencies. If $V_i \rightarrow V_j$ is a directed edge in $\mathcal{H}$, then all DAGs Markov equivalent to $\mathcal{G}$ have this directed edge; if $V_i - V_j$ is an undirected edge in $\mathcal{H}$, then some DAGs that are Markov equivalent to $\mathcal{G}$ have the directed edge $V_i \rightarrow V_j$ while others have the directed edge $V_i \leftarrow V_j$.

The PC algorithm is a well known *constraint-based*, or conditional independence based, structure learning algorithm. It is an improved greedy version of the SGS [2] and IC [3] algorithms, shown below. Instead of searching all subsets of $\mathcal{V} \backslash \{V_i, V_j\}$ for an $\mathbf{S}$ such

---

**Input**  : Observed data for variables in $\mathcal{V}$
**Output**: PDAG $\mathcal{G}$ over nodes $\mathcal{V}$

**1** $\mathcal{G} \leftarrow$ the complete undirected graph over the variables in $\mathcal{V}$
**2** For $\{V_i, V_j\} \subseteq \mathcal{V}$, if $\exists \mathbf{S} \subseteq \mathcal{V} \backslash \{V_i, V_j\}$, such that $V_i \perp\!\!\!\perp V_j \mid \mathbf{S}$, remove the $V_i - V_j$ edge
**3** For $\{V_i, V_j, V_k\} \subseteq \mathcal{V}$ such that $V_i - V_j$ and $V_j - V_k$ remain as edges, but $V_i - V_k$ does not remain, if $\not\exists \mathbf{S} \subseteq \mathcal{V} \backslash \{V_i, V_j, V_k\}$, such that $V_i \perp\!\!\!\perp V_k \mid \{\mathbf{S} \cup V_j\}$, orient $V_i \rightarrow V_j \leftarrow V_k$
**4** Orient edges to prevent additional immoralities and cycles using the Meek rules [4]

**Algorithm 1**: SGS/IC algorithm

---

that $V_i \perp\!\!\!\perp V_j \mid \mathbf{S}$, PC (i) initially sets $\mathbf{S} = \emptyset$ for all $\{V_i, V_j\}$ pairs, (ii) checks to see if any edges can be removed based on the results of conditional independence tests with these $\mathbf{S}$ sets, and (iii) iteratively increases the cardinality of $\mathbf{S}$ considered until $\not\exists V_k \in \mathcal{V}$ with degree greater than $|\mathbf{S}|$. $\mathbf{S}$ is only considered if it is a subset of nodes connected to $V_i$ or $V_j$ at the current iteration. PC learns the correct PDAG in the large sample limit when the Markov, faithfulness, and causal sufficiency (that there are no unmeasured common causes of two or more measured variables) assumptions hold [2]. The partial correlation based Fisher Z-transformation test, which assumes linear Gaussian distributions, is used for conditional independence testing with continuous variables. The statistical advantage of PC is it limits the number of tests performed, particularly those with large conditioning sets. This also yields a computational advantage since the number of possible tests is exponential in $|\mathcal{V}|$.

The recently proposed additive noise model approach to structure learning [1] assumes only that each variable can be represented as a (possibly nonlinear) function $f$ of its parents plus additive noise $\epsilon$ with some arbitrary distribution, and that the noise components are mutually independent, i.e. $\mathbb{P}(\epsilon_1, \ldots, \epsilon_n) = \prod_{i=1}^{n} \mathbb{P}(\epsilon_i)$. Consider the two variable case where $X \rightarrow Y$ is the true DAG, $X = \epsilon_X$, $Y = \sin(\pi X) + \epsilon_Y$, $\epsilon_X \sim Unif(-1, 1)$, and $\epsilon_Y \sim Unif(-1, 1)$. If we regress $Y$ on $X$ (nonparametrically), the *forward model*, figure 1a, and

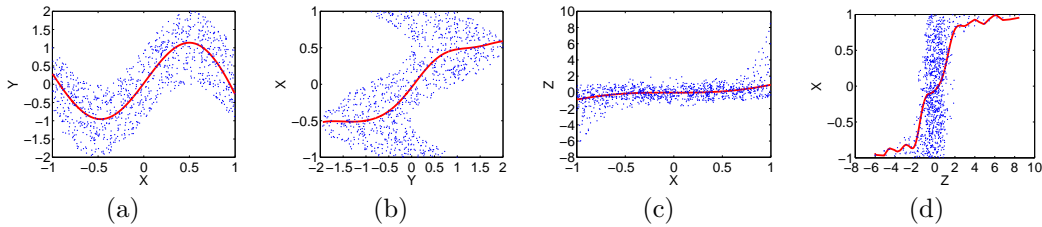

Figure 1: Nonparametric regressions with data overlayed for (a) $Y$ regressed on $X$, (b) $X$ regressed on $Y$, (c) $Z$ regressed on $X$, and (d) $X$ regressed on $Z$

regress $X$ on $Y$, the *backward model*, figure 1b, we observe the residuals $\hat{\epsilon}_Y \perp\!\!\!\perp X$ and $\hat{\epsilon}_X \not\!\perp\!\!\!\perp Y$. This provides a criterion for distinguishing $X \to Y$ from $X \leftarrow Y$ in many cases, but there are counterexamples such as the linear Gaussian case, where the forward model is invertible so we find $\hat{\epsilon}_Y \perp\!\!\!\perp X$ and $\hat{\epsilon}_X \perp\!\!\!\perp Y$. [1, 5] show, however, that whenever $f$ is nonlinear, the forward model is noninvertible, and when $f$ is linear, the forward model is only invertible when $\epsilon$ is Gaussian and a few other special cases. Another limitation of this approach is that it is not closed under marginalization of intermediary variables when $f$ is nonlinear, e.g. for $X \to Y \to Z$ with $X = \epsilon_X$, $Y = X^3 + \epsilon_Y$, $Z = Y^3 + \epsilon_Z$, $\epsilon_X \sim Unif(-1,1)$, $\epsilon_Y \sim Unif(-1,1)$, and $\epsilon_Z \sim Unif(0,1)$, observing only $X$ and $Z$, figures 1c and 1d, causes us to reject both the forward *and* backward models. [5] shows this method can be generalized to more variables. To test whether a DAG is compatible with the data, we regress each variable on its parents and test whether the resulting residuals are mutually independent. This procedure is impractical even for a few variables, however, since the number of possible DAGs grows super-exponentially with the number of variables, e.g. there are $\approx 4.2 \times 10^{18}$ DAGs with 10 nodes. Since we do not assume linearity or Gaussianity in this framework, a sufficiently powerful nonparametric independence test must be used. Typically, the Hilbert Schmidt Independence Criterion [6] is used, which we now define.

Let $X$ be a random variable with domain $\mathcal{X}$. A Hilbert space $\mathcal{H}_\mathcal{X}$ of functions from $\mathcal{X}$ to $\mathbb{R}$ is a *reproducing kernel Hilbert space* (RKHS) if for some kernel $k(\cdot, \cdot)$ (the *reproducing kernel* for $\mathcal{H}_\mathcal{X}$), for every $f(\cdot) \in \mathcal{H}_\mathcal{X}$ and $x \in \mathcal{X}$, the inner product $\langle f(\cdot), k(x, \cdot) \rangle_{\mathcal{H}_\mathcal{X}} = f(x)$. We may treat $k(x, \cdot)$ as a mapping of $x$ to the feature space $\mathcal{H}_\mathcal{X}$. For $x, x' \in \mathcal{X}$, $\langle k(x, \cdot), k(x', \cdot) \rangle_{\mathcal{H}_\mathcal{X}} = k(x, x')$, so we can compute inner products efficiently in this high dimensional space. The Moore-Aronszajn theorem shows that all symmetric positive definite kernels (most popular kernels) are reproducing kernels that uniquely define corresponding RKHSs [7]. Let $Y$ be a random variable with domain $\mathcal{Y}$ and $l(\cdot, \cdot)$ the reproducing kernel for $\mathcal{H}_\mathcal{Y}$. We define the *mean map* $\mu_X$ and *cross covariance* $\mathcal{C}_{XY}$ as follows, using $\otimes$ to denote the tensor product.

$$\mu_X = \mathbb{E}_X[k(x, \cdot)] \qquad \mathcal{C}_{XY} = ([k(x, \cdot) - \mu_X] \otimes [l(y, \cdot) - \mu_Y])$$

If the kernels are *characteristic*, e.g. Gaussian and Laplace kernels, the mean map is injective [8, 9, 10] so distinct probability distributions have different mean maps. The *Hilbert Schmidt Independence Criteria* (HSIC) $\mathbb{H}_{XY} = \|\mathcal{C}_{XY}\|^2_{HS}$ measures the dependence of $X$ and $Y$, where $\|\cdot\|_{HS}$ denotes the Hilbert Schmidt norm. [9] shows $\mathbb{H}_{XY} = 0$ if and only if $X \perp\!\!\!\perp Y$ for characteristic kernels. For $m$ paired i.i.d. samples, let $K$ and $L$ be *Gram matrices* for $k(\cdot, \cdot)$ and $l(\cdot, \cdot)$, i.e. $k_{ij} = k(x_i, x_j)$. For $H = I_N - \frac{1}{N}\mathbf{1}_N\mathbf{1}_N^T$, let $\tilde{K} = HKH$ and $\tilde{L} = HLH$ be *centered Gram matrices*. $\hat{\mathbb{H}}_{XY} = \frac{1}{m^2}tr\left(\tilde{K}\tilde{L}\right)$, where $tr$ denotes the trace, is an empirical estimator for $\mathbb{H}_{XY}$ [6]. To determine the threshold of a level-$\alpha$ statistical test, we can use the permutation approach (where we compute $\hat{\mathbb{H}}_{XY}$ for multiple random assignments of the $Y$ samples to $X$, and use the $1 - \alpha$ quantile of the resulting empirical distribution over $\hat{\mathbb{H}}_{XY}$), or a Gamma approximation to the null distribution of $m\hat{\mathbb{H}}_{XY}$ (see [6] for details).

## 3 Weakly additive noise models

We now extend the additive noise model framework to account for cases where additive noise models are invertible and cases where additive noise may not be present.

**Definition 3.1.** $\psi = \left\langle V_i, \mathbf{Pa}_{\mathcal{G}}^{V_i} \right\rangle$ is a *local additive noise model* for a distribution $\mathcal{P}$ over $\mathcal{V}$ that is Markov to a DAG $\mathcal{G} = \langle \mathcal{V}, \mathcal{E} \rangle$ if $V_i = f\left(\mathbf{Pa}_{\mathcal{G}}^{V_i}\right) + \epsilon$ is an additive noise model.

**Definition 3.2.** A *weakly additive noise model* $\mathcal{M} = \langle \mathcal{G}, \Psi \rangle$ for a distribution $\mathcal{P}$ over $\mathcal{V}$ is a DAG $\mathcal{G} = \langle \mathcal{V}, \mathcal{E} \rangle$ and set of local additive noise models $\Psi$, such that $\mathcal{P}$ is Markov to $\mathcal{G}$, $\psi \in \Psi$ if and only if $\psi$ is a local additive noise model for $\mathcal{P}$, and $\forall \left\langle V_i, \mathbf{Pa}_{\mathcal{G}}^{V_i} \right\rangle \in \Psi$, $\nexists V_j \in \mathbf{Pa}_{\mathcal{G}}^{V_i}$ such that there exists some graph $\mathcal{G}'$ (not necessarily related to $\mathcal{P}$) such that $V_i \in \mathbf{Pa}_{\mathcal{G}'}^{V_j}$ and $\left\langle V_j, \mathbf{Pa}_{\mathcal{G}'}^{V_j} \right\rangle$ is a local additive noise model for $\mathcal{P}$.

When we assume a data generating process has a weakly additive noise model representation, we assume only that there are no cases where $X \to Y$ can be written $X = f(Y) + \epsilon_X$, but not $Y = f(X) + \epsilon_Y$. In other words, the data cannot appear as though it admits an additive noise model representation, but only in the incorrect direction. This representation is still appropriate when additive noise models are invertible, and when additive noise is not present: such cases only lead to weakly additive noise models which express greater underdetermination of the true data generating process.

We now define the notion of distribution-equivalence for weakly additive noise models.

**Definition 3.3.** A weakly additive noise model $\mathcal{M} = \langle \mathcal{G}, \Psi \rangle$ is *distribution-equivalent* to $\mathcal{N} = \langle \mathcal{G}', \Psi' \rangle$ if and only if $\mathcal{G}$ and $\mathcal{G}'$ are Markov equivalent and $\psi \in \Psi$ if and only if $\psi \in \Psi'$.

Distribution-equivalence defines what can be discovered about the true data generating mechanism using observational data. We now define a new structure to partition data generating processes which instantiate distribution-equivalent weakly additive noise models.

**Definition 3.4.** A *weakly additive noise partially directed acyclic graph* (WAN-PDAG) for $\mathcal{M} = \langle \mathcal{G}, \Psi \rangle$ is a mixed graph $\mathcal{H} = \langle \mathcal{V}, \mathcal{E} \rangle$ such that for $\{V_i, V_j\} \subseteq \mathcal{V}$,

1. $V_i \to V_j$ is a directed edge in $\mathcal{H}$ if and only if $V_i \to V_j$ is a directed edge in $\mathcal{G}$ and in all $\mathcal{G}'$ such that $\mathcal{N} = \langle \mathcal{G}', \Psi' \rangle$ is distribution-equivalent to $\mathcal{M}$

2. $V_i - V_j$ is an undirected edge in $\mathcal{H}$ if and only if $V_i \to V_j$ is a directed edge in $\mathcal{G}$ and there exists a $\mathcal{G}'$ and $\mathcal{N} = \langle \mathcal{G}', \Psi' \rangle$ distribution-equivalent to $\mathcal{M}$ such that $V_i \leftarrow V_j$ is a directed edge in $\mathcal{G}'$

We now get the following results.

**Lemma 3.1.** Let $\mathcal{M} = \langle \mathcal{G}, \Psi \rangle$ be a weakly additive noise model, $\left\langle V_i, \mathbf{Pa}_{\mathcal{G}}^{V_i} \right\rangle \in \Psi$, and $\mathcal{N} = \langle \mathcal{G}', \Psi' \rangle$ be distribution equivalent to $\mathcal{M}$. Then $\mathbf{Pa}_{\mathcal{G}}^{V_i} = \mathbf{Pa}_{\mathcal{G}'}^{V_i}$ and $\mathbf{Ch}_{\mathcal{G}}^{V_i} = \mathbf{Ch}_{\mathcal{G}'}^{V_i}$.

*Proof.* Since $\mathcal{M}$ and $\mathcal{N}$ are distribution-equivalent, $\mathbf{Pa}_{\mathcal{G}}^{V_i} = \mathbf{Pa}_{\mathcal{G}'}^{V_i}$. Thus, $\mathbf{Ch}_{\mathcal{G}}^{V_i} = \mathbf{Ch}_{\mathcal{G}'}^{V_i}$ $\quad\square$

**Theorem 3.1.** The WAN-PDAG for $\mathcal{M} = \langle \mathcal{G}, \Psi \rangle$ is constructed by (i) adding all directed and undirected edges in the PDAG instantiated by $\mathcal{M}$, (ii) $\forall \left\langle V_i, \mathbf{Pa}_{\mathcal{G}}^{V_i} \right\rangle \in \Psi$, directing all $V_j \in \mathbf{Pa}_{\mathcal{G}}^{V_i}$ as $V_j \to V_i$ and all $V_k \in \mathbf{Ch}_{\mathcal{G}}^{V_i}$ as $V_i \to V_k$, and (iii) applying the extended Meek rules [4], treating orientations made using $\Psi$ as background knowledge.

*Proof.* (i) This is correct because of Markov equivalence [2]. (ii) This is correct by lemma 3.1. (iii) These rules are correct and complete [4]. $\quad\square$

WAN-PDAGs can used to identify the same information about the data generating mechanism as additive noise models, when additive noise models are identifiable, but provide a more powerful representation of uncertainty and can be used to discover more information when additive noise models are unidentifiable. The next section describes an efficient algorithm for learning WAN-PDAGs from data.

# 4 The Kernel PC (kPC) algorithm

We now describe the Kernel PC (kPC) algorithm[1], which consists of two stages: (i) a constraint-based search using the PC algorithm with a nonparametric conditional independence test (the Fisher Z test is inappropriate since we want to allow nonlinearity and non-Gaussianity) to identify the Markov equivalence class and (ii) a "PC-style" search for noninvertible additive noise models in submodels of the Markov equivalence class.

In the first stage, we use a kernel-based conditional dependence measure similar to HSIC [9] (see also [11, Section 2.2] for a related quantity with a different normalization). For a conditioning variable $Z$ with centered Gram matrix $\tilde{M}$ for a reproducing kernel $m(\cdot, \cdot)$, we define the *conditional cross covariance* $\mathcal{C}_{XY|Z} = \mathcal{C}_{\ddot{X}Z}\mathcal{C}_{ZZ}^{-1}\mathcal{C}_{Z\ddot{Y}}$, where $\ddot{X} = (X, Z)$ and $\ddot{Y} = (Y, Z)$. Let $\mathbb{H}_{XY|Z} = \|\mathcal{C}_{XY|Z}\|_{HS}^2$. It follows from [9, Theorem 3] that $\mathbb{H}_{XY|Z} = 0$ if and only if $X \perp\!\!\!\perp Y|Z$ when kernels are characteristic. [9] provides the empirical estimator:

$$\hat{\mathbb{H}}_{XY|Z} = \frac{1}{m^2}tr(\tilde{K}\tilde{L} - 2\tilde{K}\tilde{M}(\tilde{M} + \epsilon I_N)^{-2}\tilde{M}\tilde{L} + \tilde{K}\tilde{M}(\tilde{M} + \epsilon I_N)^{-2}\tilde{M}\tilde{L}\tilde{M}(\tilde{M} + \epsilon I_N)^{-2}\tilde{M})$$

The null distribution of $\hat{\mathbb{H}}_{XY|Z}$ is unknown and difficult to derive so we must use the permutation approach described in section 2. This is not straightforward since permuting $X$ or $Y$ while leaving $Z$ fixed changes the marginal distribution of $X$ given $Z$ or $Y$ given $Z$. We thus (making analogy to the discrete case) must cluster $Z$ and then permute elements only within clusters for the permutation test, as in [12].

This first stage is not computational efficient, however, since each evaluation of $\hat{\mathbb{H}}_{XY|Z}$ is naively $\mathcal{O}\left(N^3\right)$ and we need to evaluate $\hat{\mathbb{H}}_{XY|Z}$ approximately 1000 times for each permutation test. Fortunately, we see from [13, Appendix C] that the eigenspectra of Gram matrices for Gaussian kernels decay very rapidly, so low rank approximations of these matrices can be obtained even when using a very conservative threshold. We implemented the incomplete Cholesky factorization [14], which can be used to obtain an $m \times p$ matrix $G$, where $p \ll m$, and an $m \times m$ permutation matrix $P$ such that $K \approx PGG^\top P^\top$, where $K$ is an $m \times m$ Gram matrix. A clever implementation after replacing Gram matrices in $\hat{\mathbb{H}}_{XY|Z}$ with their incomplete Cholesky factorizations and using an appropriate equivalence to invert $G^\top G + \epsilon I_p$ $\left(\text{for } \tilde{M}\right)$ instead of $GG^\top + \epsilon I_m$ results in a straightforward $\mathcal{O}\left(mp^3\right)$ operation. Unfortunately, this is not numerically stable unless a relatively large regularizer $\epsilon$ is chosen or only a small number of columns are used in the incomplete Cholesky factorizations.

A more stable (and faster) approach is to obtain incomplete Cholesky factorizations $G_X, G_Y$, and $G_Z$ with permutation matrices $P_X, P_Y$, and $P_Z$, and then obtain the *thin* SVDs for $HP_XG_X, HP_YG_Y$, and $HP_ZG_Z$, e.g $HPG = USV$, where $U$ is $m \times p$, $S$ is the $p \times p$ diagonal matrix of singular values, and $V$ is $p \times p$. Now define matrices $\bar{S}^X, \bar{S}^Y$, and $\bar{S}^Z$ and $\bar{G}^X, \bar{G}^Y$, and $\bar{G}^Z$ as follows:

$$\bar{s}_{ii}^X = \left(s_{ii}^X\right)^2 \quad \bar{s}_{ii}^Y = \left(s_{ii}^Y\right)^2 \quad \bar{s}_{ii}^Z = \frac{\left(s_{ii}^Z\right)^2}{\left(s_{ii}^Z\right)^2 + \epsilon}$$

$$\bar{G}^X = U^X\bar{S}^XU^{X^\top} \quad \bar{G}^Y = U^Y\bar{S}^YU^{Y^\top} \quad \bar{G}^Z = U^Z\bar{S}^ZU^{Z^\top}$$

We can compute $\hat{\mathbb{H}}_{XY|Z} = \frac{1}{m^2}tr\left(\bar{G}^X\bar{G}^Y - 2\bar{G}^X\bar{G}^Z\bar{G}^Y + \bar{G}^X\bar{G}^Z\bar{G}^Y\bar{G}^Z\right)$ stably and efficiently in $\mathcal{O}\left(mp^3\right)$ by choosing an appropriate associative ordering of matrix multiplications. Figure 2 shows that this method leads to a significant increase in speed when used with a permutation test for conditional independence without significantly affecting the empirically observed type I error rate for a level-.05 test.

In the second stage, we look for additive noise models in submodels of the Markov equivalence class because (i) it may be more efficient to do so and require fewer tests since orientations implied by an additive noise model may imply further orientations and (ii) we

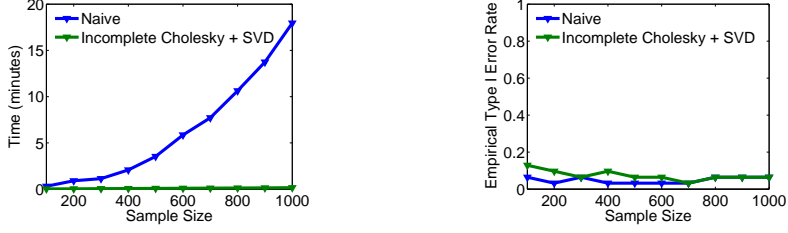

Figure 2: Runtime and Empirical Type I Error Rate. Results are over the generation of 20 3-node DAGs for which $X \perp\!\!\!\perp Y | Z$ and the generating distribution was Gaussian.

may find more orientations by considering submodels, e.g. if all relations are linear and only one variable has a non-Gaussian noise term. The basic strategy used is a "PC-style" greedy search where we look for undirected edges in the current mixed graph (starting with the PDAG resulting from the first stage) adjacent to the fewest other undirected edges. If these edges can be oriented using additive noise models, we make the implied orientations, apply the extended Meek rules, and then iterate until no more edges can be oriented. Algorithm 2 provides pseudocode. Let $\mathcal{G} = \langle \mathcal{V}, \mathcal{E} \rangle$ be the resulting PDAG and $\forall V_i \in \mathcal{V}$, let $\mathbf{U}_{\mathcal{G}}^{V_i}$ denote the nodes connected to $V_i$ in $\mathcal{G}$ by an undirected edge. We get the following results.

---

**Input** : PDAG $\mathcal{G} = \langle \mathcal{V}, \mathcal{E} \rangle$
**Output**: WAN-PDAG $\mathcal{G} = \langle \mathcal{V}, \mathcal{E} \rangle$

1   $s \leftarrow 1$

2   **while** $\max\limits_{V_i \in \mathcal{V}} \left| \boldsymbol{U}_{\mathcal{G}}^{V_i} \right| \geq s$ **do**

3     **foreach** $V_i \in \mathcal{V}$ *such that* $\left| \boldsymbol{U}_{\mathcal{G}}^{V_i} \right| = s$ *or* $\left| \boldsymbol{U}_{\mathcal{G}}^{V_i} \right| < s$ *and* $\boldsymbol{U}_{\mathcal{G}}^{V_i}$ *was updated* **do**

4       $s' \leftarrow s$

5       **while** $s' > 0$ **do**

6         **foreach** $\boldsymbol{S} \subseteq \boldsymbol{U}_{\mathcal{G}}^{V_i}$ *such that* $|\boldsymbol{S}| = s'$ *and* $\forall S_k \in \boldsymbol{S}$, *orienting* $S_k \to V_i$, *does not create an immorality* **do**

7           Nonparametrically regress $V_i$ on $\mathbf{Pa}_{\mathcal{G}}^{V_i} \cup \mathbf{S}$ and compute the residual $\hat{\epsilon}_{i\boldsymbol{S}}$

8           **if** $\hat{\epsilon}_{i\boldsymbol{S}} \perp\!\!\!\perp \boldsymbol{S}$ *and* $\nexists V_j \in \boldsymbol{S}$ *and* $\boldsymbol{S}' \subseteq \boldsymbol{U}_{\mathcal{G}}^{V_j}$ *such that. regressing* $V_j$ *on* $\boldsymbol{Pa}_{V_j}^{\mathcal{G}} \cup \boldsymbol{S}' \cup V_i$ *results in the residual* $\hat{\epsilon}_{j\boldsymbol{S}' \cup \{V_i\}} \perp\!\!\!\perp \boldsymbol{S}' \cup \{V_i\}$ **then**

9             $\forall S_k \in \mathbf{S}$, orient $S_k \to V_i$, and $\forall U_l \in \mathbf{U}_{\mathcal{G}}^{V_i} \backslash \mathbf{S}$ orient $V_i \to U_l$

10            Apply the extended Meek rules

11            $\forall V_m \in \mathcal{V}$, update $\mathbf{U}_{\mathcal{G}}^{V_m}$, set $s' = 1$, and break

12          **end**

13        **end**

14        $s' \leftarrow s' - 1$;

15       **end**

16     **end**

17     $s \leftarrow s + 1$

18 **end**

**Algorithm 2**: Second Stage of kPC

---

**Lemma 4.1.** If an edge is oriented in the second stage of kPC, it is implied by a noninvertible local additive noise model.

*Proof.* If the condition at line 8 is true then $\left\langle V_i, \mathbf{Pa}_{\mathcal{G}}^{V_i} \cup \mathbf{S} \right\rangle$ is a noninvertible local additive noise model. All $U_l \in \mathbf{U}_{\mathcal{G}}^{V_i} \backslash \mathbf{S}$ must be children of $V_i$ by lemma 3.1. $\qquad \square$

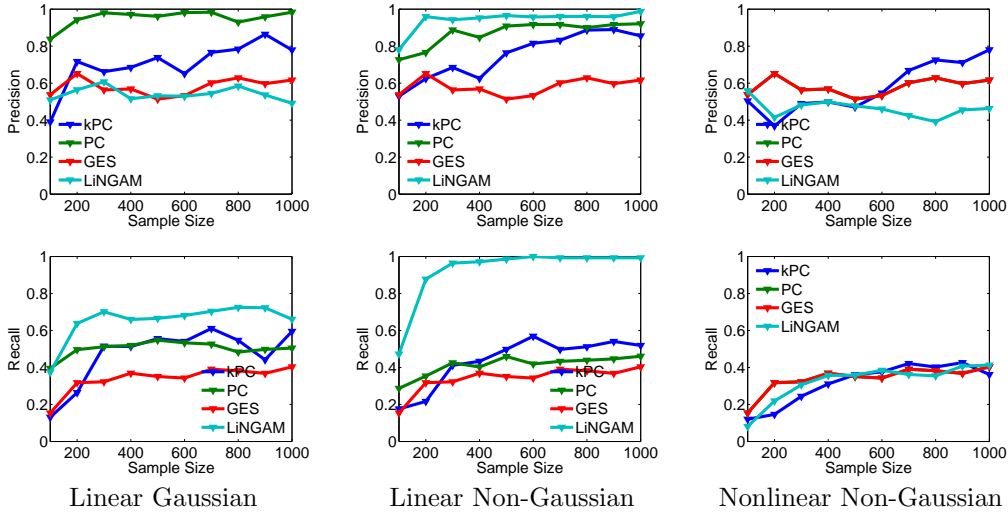

Figure 3: Precision and Recall

**Lemma 4.2.** Suppose $\psi = \langle V_i, \mathbf{W} \rangle$ is a noninvertible local additive noise model. Then kPC will make all orientations implied by $\psi$.

*Proof.* Let $\tilde{\mathbf{S}} = \mathbf{W} \backslash \mathbf{Pa}_{V_i}^{\mathcal{G}}$ for $\mathbf{Pa}_{V_i}^{\mathcal{G}}$ at the current iteration. kPC must terminate with $s > |\tilde{\mathbf{S}}|$ since $|\tilde{\mathbf{S}}| \leq |\mathbf{U}_{\mathcal{G}}^{V_i}|$ so $\mathbf{S} = \tilde{\mathbf{S}}$ at some iteration. Since $\langle V_i, \mathbf{Pa}_{\mathcal{G}}^{V_i} \cup \tilde{\mathbf{S}} \rangle$ is a noninvertible local additive noise model, line 8 is satisfied so all edges connected to $V_i$ are oriented. □

**Theorem 4.1.** Assume data is generated according to some weakly additive noise model $\mathcal{M} = \langle \mathcal{G}, \Psi \rangle$. Then kPC will return the WAN-PDAG instantiated by $\mathcal{M}$ assuming perfect conditional independence information, Markov, faithfulness, and causal sufficiency.

*Proof.* The PC algorithm is correct and complete with respect to conditional independence [2]. Orientations made with respect to additive noise models are correct by lemma 4.1 and all such orientations that can be made are made by lemma 4.2. The Meek rules, which are correct and complete [4], are invoked after each orientation made with respect to additive noise models so they are invoked after all such orientations are made. □

## 5  Related research

kPC is similar in spirit to the PC-LiNGAM structure learning algorithm [15], which assumes dependencies are linear with either Gaussian or non-Gaussian noise. PC-LiNGAM combines the PC algorithm with LiNGAM to learn structures referred to as *ngDAG*s. KCL [11] is a heuristic search for a mixed graph that uses the same kernel-based dependence measures as kPC (while not determining significance threshholds via a hypothesis test), but does not take advantage of additive noise models. [16] provides a more efficient algorithm for learning additive noise models, by first finding a causal ordering after doing a series of high dimensional regressions and HSIC independence tests and then pruning the resulting DAG implied by this ordering. Finally, [17] proposes a two-stage procedure for learning additive noise models from data that is similar to kPC, but requires the additive noise model assumptions in the first stage where the Markov equivalence class is identified.

## 6  Experimental results

To evaluate kPC, we generated 20 random 7-nodes DAGs using the MCMC algorithm in [18] and sampled 1000 data points from each DAG under three conditions: linear dependencies

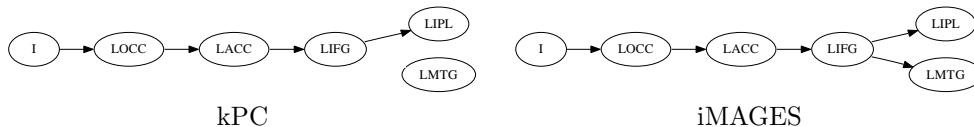

$$\text{kPC} \qquad\qquad\qquad \text{iMAGES}$$

Figure 4: Structures learned by kPC and iMAGES

with Gaussian noise, linear dependencies with non-Gaussian noise, and nonlinear dependencies with non-Gaussian noise. We generated non-Gaussian noise using the same procedure as [19] and used polynomial and trigonometric functions for nonlinear dependencies.

We compared kPC to PC, the score-based GES with the BIC-score [20], and the ICA-based LiNGAM [19], which assumes linear dependencies and non-Gaussian noise. We applied two metrics in measuring performance vs sample size: precision, i.e. proportion of directed edges in the resulting graph that are in the true DAG, and recall, i.e. proportion of directed edges in the true DAG that are in the resulting graph. Figure 3 reports the results. In the linear Gaussian case, we see PC shows slightly better performance than kPC in precision, which is unsurprising since PC assumes linear Gaussian distributions. Only LiNGAM shows better recall, but worse precision. LiNGAM performs significantly better than the other algorithms in the linear non-Gaussian case. kPC performs about the same as PC in precision and recall, which again is unsurprising since previous simulation results have shown that nonlinearity, but not non-Gaussianity can significantly affect the performance of PC. In the nonlinear non-Gaussian case, kPC performs slightly better than PC in precision. We note, however, that in some of these cases the performance of kPC was significantly better.[2]

We also ran kPC on data from an fMRI experiment that is analyzed in [21] where nonlinear dependencies can be observed. Figure 4 shows the structure that kPC learned, where each of the nodes corresponds to a particular brain region. This structure is the same as the one learned by the (GES-style) iMAGES algorithm in [21] except for the absence of one edge. However, iMAGES required background knowledge to direct the edges. kPC successfully found the same directed edges without using any background knowledge. Domain experts in neuroscience have confirmed the plausibility of the observed relationships.

# 7    Conclusion

We introduced weakly additive noise models, which extend the additive noise model framework to cases such as the linear Gaussian, where the additive noise model is invertible and thus unidentifiable, as well as cases where additive noise is not present. The weakly additive noise framework allows us to identify a unique DAG when the additive noise model assumptions hold, and a structure that is at least as specific as a PDAG (possibly still a unique DAG) when some additive noise assumptions fail. We defined equivalence classes for such models and introduced the kPC algorithm for learning these equivalence classes from data. Finally, we found that the algorithm performed well on both synthetic and real data.

## Acknowledgements

We thank Dominik Janzing and Bernhard Schölkopf for helpful comments. RET was funded by a grant from the James S. McDonnel Foundation. AG was funded by DARPA IPTO FA8750-09-1-0141, ONR MURI N000140710747, and ARO MURI W911NF0810242.

## Footnotes

[1]MATLAB code may be obtained from http://www.andrew.cmu.edu/~rtillman/kpc

[2]When simulating nonlinear data, we must be careful to ensure that variances do not blow up and result in data for which no finite sample method can show adequate performance. This has the unfortunate side effect that the nonlinear data generated may be well approximated using linear methods. Future research will consider more sophisticated methods for simulating data that is more appropriate when comparing kPC to linear methods.

# References

[1] P. O. Hoyer, D. Janzing, J. M. Mooij, J. Peters, and B. Schölkopf. Nonlinear causal discovery with additive noise models. In *Advances in Neural Information Processing Systems 21*, 2009.

[2] P. Spirtes, C. Glymour, and R. Scheines. *Causation, Prediction, and Search*. 2nd edition, 2000.

[3] J. Pearl. *Causality: Models, Reasoning, and Inference*. 2000.

[4] C. Meek. Causal inference and causal explanation with background knowledge. In *Proceedings of the 11th Conference on Uncertainty in Artificial Intelligence*, 1995.

[5] K. Zhang and A. Hyvärinen. On the identifiability of the post-nonlinear causal model. In *Proceedings of the 26th Conference on Uncertainty in Artificial Intelligence*, 2009.

[6] A. Gretton, K. Fukumizu, C. H. Teo, L. Song, B. Schölkopf, and A. J. Smola. A kernel statistical test of independence. In *Advances in Neural Information Processing Systems 20*, 2008.

[7] Nachman Aronszajn. Theory of reproducing kernels. *Transactions of the American Mathematical Society*, 68(3):337404, 1950.

[8] A. Gretton, K. Borgwardt, M. Rasch, B. Schölkopf, and A. Smola. A kernel method for the two-sample-problem. In *Advances in Neural Information Processing Systems 19*, 2007.

[9] K. Fukumizu, A. Gretton, X. Sun, and B. Schölkopf. Kernel measures of conditional dependence. In *Advances in Neural Information Processing Systems 20*, 2008.

[10] B. Sriperumbudur, A. Gretton, K. Fukumizu, G. Lanckriet, and B. Schölkopf. Injective hilbert space embeddings of probability measures. In *Proceedings of the 21st Annual Conference on Learning Theory*, 2008.

[11] X. Sun, D. Janzing, B. Scholköpf, and K. Fukumizu. A kernel-based causal learning algorithm. In *Proceedings of the 24th International Conference on Machine Learning*, 2007.

[12] X. Sun. *Causal inference from statistical data*. PhD thesis, Max Plank Institute for Biological Cybernetics, 2008.

[13] F. R. Bach and M. I. Jordan. Kernel independent component analysis. *Journal of Machine Learning Research*, 3:1–48, 2002.

[14] S. Fine and K. Scheinberg. Efficient SVM training using low-rank kernel representations. *Journal of Machine Learning Research*, 2:243–264, 2001.

[15] P. O. Hoyer, A. Hyvärinen, R. Scheines, P. Spirtes, J. Ramsey, G. Lacerda, and S. Shimizu. Causal discovery of linear acyclic models with arbitrary distributions. In *Proceedings of the 24th Conference on Uncertainty in Artificial Intelligence*, 2008.

[16] J. M. Mooij, D. Janzing, J. Peters, and B. Scholköpf. Regression by dependence minimization and its application to causal inference in additive noise models. In *Proceedings of the 26th International Conference on Machine Learning*, 2009.

[17] K. Zhang and A. Hyvärinen. Acyclic causality discovery with additive noise: An information-theoretical perspective. In *Proceedings of the European Conference on Machine Learning and Principles and Practice of Knowledge Discovery in Databases 2009*, 2009.

[18] G. Melançon, I. Dutour, and M. Bousquet-Mélou. Random generation of dags for graph drawing. Technical Report INS-R0005, Centre for Mathematics and Computer Sciences, 2000.

[19] S. Shimizu, P. Hoyer, A. Hyvärinen, and A. Kerminen. A linear non-gaussian acyclic model for causal discovery. *Journal of Machine Learning Research*, 7:1003–2030, 2006.

[20] D. M. Chickering. Optimal structure identification with greedy search. *Journal of Machine Learning Research*, 3:507–554, 2002.

[21] J. D. Ramsey, S. J. Hanson, C. Hanson, Y. O. Halchenko, R. A. Poldrack, and C. Glymour. Six problems for causal inference from fMRI. *NeuroImage*, 2009. In press.

